# Practical Variational Inference for Neural Networks

**Alex Graves**
Department of Computer Science
University of Toronto, Canada
graves@cs.toronto.edu

## Abstract

Variational methods have been previously explored as a tractable approximation to Bayesian inference for neural networks. However the approaches proposed so far have only been applicable to a few simple network architectures. This paper introduces an easy-to-implement stochastic variational method (or equivalently, minimum description length loss function) that can be applied to most neural networks. Along the way it revisits several common regularisers from a variational perspective. It also provides a simple pruning heuristic that can both drastically reduce the number of network weights and lead to improved generalisation. Experimental results are provided for a hierarchical multidimensional recurrent neural network applied to the TIMIT speech corpus.

## 1 Introduction

In the eighteen years since variational inference was first proposed for neural networks [10] it has not seen widespread use. We believe this is largely due to the difficulty of deriving analytical solutions to the required integrals over the variational posteriors. Such solutions are complicated for even the simplest network architectures, such as radial basis networks [2] and single layer feedforward networks with linear outputs [10, 1, 14], and are generally unavailable for more complex systems.

The approach taken here is to forget about analytical solutions and search instead for variational distributions whose expectation values (and derivatives thereof) can be efficiently approximated with numerical integration. While it may seem perverse to replace one intractable integral (over the true posterior) with another (over the variational posterior), the point is that the variational posterior is far easier to draw probable samples from, and correspondingly more amenable to numerical methods. The result is a stochastic method for variational inference with a diagonal Gaussian posterior that can be applied to any differentiable log-loss parametric model—which includes most neural networks[1]

Variational inference can be reformulated as the optimisation of a *Minimum Description length* (MDL; [21]) loss function; indeed it was in this form that variational inference was first considered for neural networks. One advantage of the MDL interpretation is that it leads to a clear separation between prediction accuracy and model complexity, which can help to both analyse and optimise the network. Another benefit is that recasting inference as optimisation makes it to easier to implement in existing, gradient-descent-based neural network software.

## 2 Neural Networks

For the purposes of this paper a neural network is a parametric model that assigns a conditional probability $\Pr(\mathcal{D}|\mathbf{w})$ to some dataset $\mathcal{D}$, given a set $\mathbf{w} = \{w_i\}_{i=1}^{W}$ of real-valued parameters, or *weights*. The elements $(\mathbf{x}, \mathbf{y})$ of $\mathcal{D}$, each consisting of an input $\mathbf{x}$ and a target $\mathbf{y}$, are assumed to be

drawn independently from a joint distribution $p(\mathbf{x}, \mathbf{y})$[2]. The *network loss* $L^N(\mathbf{w}, \mathcal{D})$ is defined as the negative log probability of the data given the weights.

$$L^N(\mathbf{w}, \mathcal{D}) = -\ln \Pr(\mathcal{D}|\mathbf{w}) = -\sum_{(\mathbf{x}, \mathbf{y}) \in \mathcal{D}} \ln \Pr(\mathbf{y}|\mathbf{x}, \mathbf{w}) \tag{1}$$

The logarithm could be taken to any base, but to avoid confusion we will use the natural logarithm $\ln$ throughout. We assume that the partial derivatives of $L^N(\mathbf{w}, \mathcal{D})$ with respect to the network weights can be efficiently calculated (using, for example, backpropagation or backpropagation through time [22]).

## 3  Variational Inference

Performing Bayesian inference on a neural network requires the posterior distribution of the network weights given the data. If the weights have a prior probability $P(\mathbf{w}|\boldsymbol{\alpha})$ that depends on some parameters $\boldsymbol{\alpha}$, the posterior can be written $\Pr(\mathbf{w}|\mathcal{D}, \boldsymbol{\alpha})$. Unfortunately, for most neural networks $\Pr(\mathbf{w}|\mathcal{D}, \boldsymbol{\alpha})$ cannot be calculated analytically, or even efficiently sampled from. Variational inference addresses this problem by approximating $\Pr(\mathbf{w}|\mathcal{D}, \boldsymbol{\alpha})$ with a more tractable distribution $Q(\mathbf{w}|\boldsymbol{\beta})$. The approximation is fitted by minimising the *variational free energy* $\mathcal{F}$ with respect to the parameters $\boldsymbol{\beta}$, where

$$\mathcal{F} = -\left\langle \ln \left[ \frac{\Pr(\mathcal{D}|\mathbf{w})P(\mathbf{w}|\boldsymbol{\alpha})}{Q(\mathbf{w}|\boldsymbol{\beta})} \right] \right\rangle_{\mathbf{w} \sim Q(\boldsymbol{\beta})} \tag{2}$$

and for some function $g$ of a random variable $x$ with distribution $p(x)$, $\langle g \rangle_{x \sim p}$ denotes the expectation of $g$ over $p$. A fully Bayesian approach would infer the prior parameters $\boldsymbol{\alpha}$ from a hyperprior; however in this paper they are found by simply minimising $\mathcal{F}$ with respect to $\boldsymbol{\alpha}$ as well as $\boldsymbol{\beta}$.

## 4  Minimum Description Length

$\mathcal{F}$ can be reinterpreted as a minimum description length loss function [12] by rearranging Eq. (2) and substituting in from Eq. (1) to get

$$\mathcal{F} = \left\langle L^N(\mathbf{w}, \mathcal{D}) \right\rangle_{\mathbf{w} \sim Q(\boldsymbol{\beta})} + D_{KL}(Q(\boldsymbol{\beta})||P(\boldsymbol{\alpha})), \tag{3}$$

where $D_{KL}(Q(\boldsymbol{\beta})||P(\boldsymbol{\alpha}))$ is the Kullback-Leibler divergence between $Q(\boldsymbol{\beta})$ and $P(\boldsymbol{\alpha})$. Shannon's source coding theorem [23] tells us that the first term on the right hand side of Eq. (3) is a lower bound on the expected amount of information (measured in *nats*, due to the use of natural logarithms) required to transmit the targets in $\mathcal{D}$ to a receiver who knows the inputs, using the outputs of a network whose weights are sampled from $Q(\boldsymbol{\beta})$. Since this term decreases as the network's prediction accuracy increases, we identify it as the *error loss* $L^E(\boldsymbol{\beta}, \mathcal{D})$:

$$L^E(\boldsymbol{\beta}, \mathcal{D}) = \left\langle L^N(\mathbf{w}, \mathcal{D}) \right\rangle_{\mathbf{w} \sim Q(\boldsymbol{\beta})} \tag{4}$$

Shannon's bound can almost be achieved in practice using arithmetic coding [26]. The second term on the right hand side of Eq. (3) is the expected number of nats required by a receiver who knows $P(\boldsymbol{\alpha})$ to pick a sample from $Q(\boldsymbol{\beta})$. Since this term measures the cost of 'describing' the network weights to the receiver, we identify it as the *complexity loss* $L^C(\boldsymbol{\alpha}, \boldsymbol{\beta})$:

$$L^C(\boldsymbol{\alpha}, \boldsymbol{\beta}) = D_{KL}(Q(\boldsymbol{\beta})||P(\boldsymbol{\alpha})) \tag{5}$$

$L^C(\boldsymbol{\alpha}, \boldsymbol{\beta})$ can be realised with *bits-back* coding [25, 10]. Although originally conceived as a thought experiment, bits-back coding has been used for an actual compression scheme [5]. Putting the terms together $\mathcal{F}$ can be rephrased as an MDL loss function $L(\boldsymbol{\alpha}, \boldsymbol{\beta}, \mathcal{D})$ that measures the total number of nats required to transmit the training targets using the network, given $\boldsymbol{\alpha}$ and $\boldsymbol{\beta}$:

$$L(\boldsymbol{\alpha}, \boldsymbol{\beta}, \mathcal{D}) = L^E(\boldsymbol{\beta}, \mathcal{D}) + L^C(\boldsymbol{\alpha}, \boldsymbol{\beta}) \tag{6}$$

The network is then trained on $\mathcal{D}$ by minimising $L(\boldsymbol{\alpha}, \boldsymbol{\beta}, \mathcal{D})$ with respect to $\boldsymbol{\alpha}$ and $\boldsymbol{\beta}$, just like an ordinary neural network loss function. One advantage of using a transmission cost as a loss

function is that we can immediately determine whether the network has compressed the targets past a reasonable benchmark (such as that given by an off-the-shelf compressor). If it has, we can be fairly certain that the network is learning underlying patterns in the data and not simply memorising the training set. We would therefore expect it to generalise well to new data. In practice we have found that as long as significant compression is taking place, decreasing $L(\boldsymbol{\alpha}, \boldsymbol{\beta}, \mathcal{D})$ on the training set does not increase $L^E(\boldsymbol{\beta}, \mathcal{D})$ on the test set, and it is therefore unnecessary to sacrifice any training data for early stopping.

Two transmission costs were ignored in the above discussion. One is the cost of transmitting the model with $\mathbf{w}$ unspecified (for example software that implements the network architecture, the training algorithm etc.). The other is the cost of transmitting the prior. If either of these are used to encode a significant amount of information about $\mathcal{D}$, the MDL principle will break down and the generalisation guarantees that come with compression will be lost. The easiest way to prevent this is to keep both costs very small compared to $\mathcal{D}$. In particular the prior should not contain too many parameters.

## 5    Choice of Distributions

We now derive the form of $L^E(\boldsymbol{\beta}, \mathcal{D})$ and $L^C(\boldsymbol{\alpha}, \boldsymbol{\beta})$ for various choices of $Q(\boldsymbol{\beta})$ and $P(\boldsymbol{\alpha})$. We also derive the gradients of $L^E(\boldsymbol{\beta}, \mathcal{D})$ and $L^C(\boldsymbol{\alpha}, \boldsymbol{\beta})$ with respect to $\boldsymbol{\beta}$ and the optimal values of $\boldsymbol{\alpha}$ given $\boldsymbol{\beta}$. All continuous distributions are implicitly assumed to be quantised at some very fine resolution, and we will limit ourselves to diagonal posteriors of the form $Q(\boldsymbol{\beta}) = \prod_{i=1}^{W} q_i(\beta_i)$, meaning that $L^C(\boldsymbol{\alpha}, \boldsymbol{\beta}) = \sum_{i=1}^{W} D_{KL}(q_i(\beta_i)||P(\boldsymbol{\alpha}))$.

### 5.1    Delta Posterior

Perhaps the simplest nontrivial distribution for $Q(\boldsymbol{\beta})$ is a *delta distribution* that assigns probability 1 to a particular set of weights $\mathbf{w}$ and 0 to all other weights. In this case $\boldsymbol{\beta} = \mathbf{w}$, $L^E(\boldsymbol{\beta}, \mathcal{D}) = L^N(\mathbf{w}, \mathcal{D})$ and $L^C(\boldsymbol{\alpha}, \boldsymbol{\beta}) = L^C(\boldsymbol{\alpha}, \mathbf{w}) = -logP(\mathbf{w}|\boldsymbol{\alpha}) + C$. where $C$ is a constant that depends only on the discretisation of $Q(\boldsymbol{\beta})$. Although $C$ has no effect on the gradient used for training, it is usually large enough to ensure that the network cannot compress the data using the coding scheme described in the previous section[3]. If the prior is uniform, and all realisable weight values are equally likely then $L^C(\boldsymbol{\alpha}, \boldsymbol{\beta})$ is a constant and we recover ordinary maximum likelihood training.

If the prior is a Laplace distribution then $\boldsymbol{\alpha} = \{\mu, b\}$, $P(\mathbf{w}|\boldsymbol{\alpha}) = \prod_{i=1}^{W} \frac{1}{2b} \exp\left(-\frac{|w_i - \mu|}{b}\right)$ and

$$L^C(\boldsymbol{\alpha}, \mathbf{w}) = W \ln 2b + \frac{1}{b} \sum_{i=1}^{W} |w_i - \mu| + C \implies \frac{\partial L^C(\boldsymbol{\alpha}, \mathbf{w})}{\partial w_i} = \frac{sgn(w_i - \mu)}{b} \qquad (7)$$

If $\mu = 0$ and $b$ is fixed, this is equivalent to ordinary L1 regularisation. However we can instead determine the optimal prior parameters $\hat{\boldsymbol{\alpha}}$ for $\mathbf{w}$ as follows: $\hat{\mu} = \mu_{1/2}(\mathbf{w})$ (the median weight value) and $\hat{b} = \frac{1}{W} \sum_{i=1}^{W} |w_i - \hat{\mu}|$.

If the prior is Gaussian then $\boldsymbol{\alpha} = \{\mu, \sigma^2\}$, $P(\mathbf{w}|\boldsymbol{\alpha}) = \prod_{i=1}^{W} \frac{1}{\sqrt{2\pi\sigma^2}} \exp\left(-\frac{(w_i - \mu)^2}{2\sigma^2}\right)$ and

$$L^C(\boldsymbol{\alpha}, \mathbf{w}) = W \ln(\sqrt{2\pi\sigma^2}) + \frac{1}{2\sigma^2} \sum_{i=1}^{W} (w_i - \mu)^2 + C \implies \frac{\partial L^C(\boldsymbol{\alpha}, \mathbf{w})}{\partial w_i} = \frac{w_i - \mu}{\sigma^2} \qquad (8)$$

With $\mu = 0$ and $\sigma^2$ fixed this is equivalent to L2 regularisation (also known as *weight decay* for neural networks). The optimal $\hat{\boldsymbol{\alpha}}$ given $\mathbf{w}$ are $\hat{\mu} = \frac{1}{W} \sum_{i=1}^{W} w_i$ and $\hat{\sigma}^2 = \frac{1}{W} \sum_{i=1}^{W} (w_i - \hat{\mu})^2$

### 5.2    Gaussian Posterior

A more interesting distribution for $Q(\boldsymbol{\beta})$ is a diagonal Gaussian. In this case each weight requires a separate mean and variance, so $\boldsymbol{\beta} = \{\boldsymbol{\mu}, \boldsymbol{\sigma^2}\}$ with the mean vector $\boldsymbol{\mu}$ and variance vector $\boldsymbol{\sigma^2}$ both

the same size as $\mathbf{w}$. For a general network architecture we cannot compute either $L^E(\boldsymbol{\beta}, \mathcal{D})$ or its derivatives exactly, so we resort to sampling. Applying Monte-Carlo integration to Eq. (4) gives

$$L^E(\boldsymbol{\beta}, \mathcal{D}) \approx \frac{1}{S} \sum_{k=1}^{S} L^N(\mathbf{w}^k, \mathcal{D}) \tag{9}$$

with $\mathbf{w}^k$ drawn independently from $Q(\boldsymbol{\beta})$. A combination of the Gaussian characteristic function and integration by parts can be used to derive the following identities for the derivatives of multivariate Gaussian expectations [18]:

$$\nabla_{\boldsymbol{\mu}} \langle V(\boldsymbol{a}) \rangle_{\boldsymbol{a} \sim \mathcal{N}} = \langle \nabla_{\boldsymbol{a}} V(\boldsymbol{a}) \rangle_{\boldsymbol{a} \sim \mathcal{N}}, \qquad \nabla_{\boldsymbol{\Sigma}} \langle V(\boldsymbol{a}) \rangle_{\boldsymbol{a} \sim \mathcal{N}} = \frac{1}{2} \langle \nabla_{\boldsymbol{a}} \nabla_{\boldsymbol{a}} V(\boldsymbol{a}) \rangle_{\boldsymbol{a} \sim \mathcal{N}} \tag{10}$$

where $\mathcal{N}$ is a multivariate Gaussian with mean vector $\boldsymbol{\mu}$ and covariance matrix $\boldsymbol{\Sigma}$, and $V$ is an arbitrary function of $\boldsymbol{a}$. Differentiating Eq. (4) and applying these identities yields

$$\frac{\partial L^E(\boldsymbol{\beta}, \mathcal{D})}{\partial \mu_i} = \left\langle \frac{\partial L^N(\mathbf{w}, \mathcal{D})}{\partial w_i} \right\rangle_{\mathbf{w} \sim Q(\boldsymbol{\beta})} \approx \frac{1}{S} \sum_{k=1}^{S} \frac{\partial L^N(\mathbf{w}^k, \mathcal{D})}{\partial w_i} \tag{11}$$

$$\frac{\partial L^E(\boldsymbol{\beta}, \mathcal{D})}{\partial \sigma_i^2} = \frac{1}{2} \left\langle \frac{\partial^2 L^N(\mathbf{w}, \mathcal{D})}{\partial w_i^2} \right\rangle_{\mathbf{w} \sim Q(\boldsymbol{\beta})} \approx \frac{1}{2} \left\langle \left[ \frac{\partial L^N(\mathbf{w}, \mathcal{D})}{\partial w_i} \right]^2 \right\rangle_{\mathbf{w} \sim Q(\boldsymbol{\beta})} \approx \frac{1}{2S} \sum_{k=1}^{S} \left[ \frac{\partial L^N(\mathbf{w}^k, \mathcal{D})}{\partial w_i} \right]^2 \tag{12}$$

where the first approximation in Eq. (12) comes from substituting the negative diagonal of the empirical Fisher information matrix for the diagonal of the Hessian. This approximation is exact if the conditional distribution $\Pr(\mathcal{D}|\mathbf{w})$ matches the empirical distribution of $\mathcal{D}$ (i.e. if the network perfectly models the data); we would therefore expect it to improve as $L^E(\boldsymbol{\beta}, \mathcal{D})$ decreases. For simple networks whose second derivatives can be calculated efficiently the approximation is unnecessary and the diagonal Hessian can be sampled instead.

A simplification of the above distribution is to consider the variances of $Q(\boldsymbol{\beta})$ fixed and optimise only the means. Then the sampling used to calculate the derivatives in Eq. (11) is equivalent to adding zero-mean, fixed-variance Gaussian noise to the network weights during training. In particular, if the prior $P(\boldsymbol{\alpha})$ is uniform and a single weight sample is taken for each element of $\mathcal{D}$, then minimising $L(\boldsymbol{\alpha}, \boldsymbol{\beta}, \mathcal{D})$ is identical to minimising $L^N(\mathbf{w}, \mathcal{D})$ with *weight noise* or *synaptic noise* [13]. Note that the quantisation of the uniform prior adds a large constant to $L^C(\boldsymbol{\alpha}, \boldsymbol{\beta})$, making it unfeasible to compress the data with our MDL coding scheme; in practice early stopping is required to prevent overfitting when training with weight noise.

If the prior is Gaussian then $\boldsymbol{\alpha} = \{\mu, \sigma^2\}$ and

$$L^C(\boldsymbol{\alpha}, \boldsymbol{\beta}) = \sum_{i=1}^{W} \ln \frac{\sigma}{\sigma_i} + \frac{1}{2\sigma^2} \left[ (\mu_i - \mu)^2 + \sigma_i^2 - \sigma^2 \right] \tag{13}$$

$$\implies \frac{\partial L^C(\boldsymbol{\alpha}, \boldsymbol{\beta})}{\partial \mu_i} = \frac{\mu_i - \mu}{\sigma^2}, \qquad \frac{\partial L^C(\boldsymbol{\alpha}, \boldsymbol{\beta})}{\partial \sigma_i^2} = \frac{1}{2} \left[ \frac{1}{\sigma^2} - \frac{1}{\sigma_i^2} \right] \tag{14}$$

The optimal prior parameters $\hat{\boldsymbol{\alpha}}$ given $\boldsymbol{\beta}$ are

$$\hat{\mu} = \frac{1}{W} \sum_{i=1}^{W} \mu_i, \qquad \hat{\sigma}^2 = \frac{1}{W} \sum_{i=1}^{W} \left[ \sigma_i^2 + (\mu_i - \hat{\mu})^2 \right] \tag{15}$$

If a Gaussian prior is used with the fixed variance 'weight noise' posterior described above, it is still possible to choose the optimal prior parameters for each $\boldsymbol{\beta}$. This requires only a slight modification of standard weight-noise training, with the derivatives on the left of Eq. (14) added to the weight gradient and $\boldsymbol{\alpha}$ optimised after every weight update. But because the prior is no longer uniform the network is able to compress the data, making it feasible to dispense with early stopping.

The terms in the sum on the right hand side of Eq. (13) are the complexity costs of individual network weights. These costs give valuable insight into the internal structure of the network, since (with a limited budget of bits to spend) the network will assign more bits to more important weights. Importance can be used, for example, to prune away spurious weights [15] or determine which inputs are relevant [16].

# 6 Optimisation

If the derivatives of $L^E(\boldsymbol{\beta}, \mathcal{D})$ are stochastic, we require an optimiser that can tolerate noisy gradient estimates. Steepest descent with momentum [19] and RPROP [20] both work well in practice.

Although stochastic derivatives should in principle be estimated using the same weight samples for the entire dataset, it is in practice much more efficient to pick different weight samples for each $(\mathbf{x}, \mathbf{y}) \in \mathcal{D}$. If both the prior and posterior are Gaussian this yields

$$\frac{\partial L(\boldsymbol{\alpha}, \boldsymbol{\beta}, \mathcal{D})}{\partial \mu_i} \approx \frac{\mu_i - \mu}{\sigma^2} + \sum_{(\mathbf{x}, \mathbf{y}) \in \mathcal{D}} \frac{1}{S} \sum_{k=1}^{S} \frac{\partial L^N(\mathbf{w}^k, \mathbf{x}, \mathbf{y})}{\partial w_i} \tag{16}$$

$$\frac{\partial L(\boldsymbol{\alpha}, \boldsymbol{\beta}, \mathcal{D})}{\partial \sigma_i^2} \approx \frac{1}{2} \left[ \frac{1}{\sigma^2} - \frac{1}{\sigma_i^2} \right] + \sum_{(\mathbf{x}, \mathbf{y}) \in \mathcal{D}} \frac{1}{2S} \sum_{k=1}^{S} \left[ \frac{\partial L^N(\mathbf{w}^k, \mathbf{x}, \mathbf{y})}{\partial w_i} \right]^2 \tag{17}$$

where $L^N(\mathbf{w}^k, \mathbf{x}, \mathbf{y}) = -\ln \Pr(\mathbf{y}|\mathbf{x}, \mathbf{w})$ and a separate set of $S$ weight samples $\{\mathbf{w}^k\}_{k=1}^{S}$ is drawn from $Q(\boldsymbol{\beta})$ for each $(\mathbf{x}, \mathbf{y})$. For large datasets it is usually sufficient to set $S = 1$; however performance can in some cases be substantially improved by using more samples, at the cost of longer training times.

If the data is divided into $B$ equally-sized batches such that $\mathcal{D} = \{b_j\}_{j=1}^{B}$, and an 'online' optimiser is used, with the parameters updated after each batch gradient calculation, the following online loss function (and corresponding derivatives) should be employed:

$$L(\boldsymbol{\alpha}, \boldsymbol{\beta}, b_j) = \frac{1}{B} L^C(\boldsymbol{\alpha}, \boldsymbol{\beta}) + L^E(\boldsymbol{\beta}, b_j) \tag{18}$$

Note the $1/B$ factor for the complexity loss. This is because the weights (to which the complexity cost applies) are only transmitted once for the entire dataset, whereas the error cost must be transmitted separately for each batch.

During training, the prior parameters $\boldsymbol{\alpha}$ should be set to their optimal values after every update to $\boldsymbol{\beta}$. For more complex priors where the optimal $\boldsymbol{\alpha}$ cannot be found in closed form (such as mixture distributions), $\boldsymbol{\alpha}$ and $\boldsymbol{\beta}$ can instead be optimised simultaneously with gradient descent [17, 10].

Ideally a trained network should be evaluated on some previously unseen input $\mathbf{x}'$ using the expected distribution $\langle \Pr(.|\mathbf{x}', \mathbf{w}) \rangle_{\mathbf{w} \sim Q(\boldsymbol{\beta})}$. However the *maximum a posteriori* approximation $\Pr(.|\mathbf{x}', \mathbf{w}^*)$, where $\mathbf{w}^*$ is the mode of $Q(\boldsymbol{\beta})$, appears to work well in practice (at least for diagonal Gaussian posteriors). This is equivalent to removing weight noise during testing.

# 7 Pruning

Removing weights from a neural network (a process usually referred to as *pruning*) has been repeatedly proposed as a means of reducing complexity and thereby improving generalisation [15, 7]. This would seem redundant for variational inference, which automatically limits the network complexity. However pruning can reduce the computational cost and memory demands of the network. Furthermore we have found that if the network is retrained after pruning, the final performance can be improved. A possible explanation is that pruning reduces the noise in the gradient estimates (because the pruned weights are not sampled) without increasing network complexity.

Weights $\mathbf{w}$ that are more probable under $Q(\boldsymbol{\beta})$ tend to give lower $L^N(\mathbf{w}, \mathcal{D})$ and pruning a weight is equivalent to fixing it to zero. These two facts suggest a pruning heuristic where a weight is removed if its probability density at zero is sufficiently high under $Q(\boldsymbol{\beta})$. For a diagonal posterior we can define the *relative* probability of each $w_i$ at zero as the density of $q_i(\beta_i)$ at zero divided by the density of $q_i(\beta_i)$ at its mode. We can then define a pruning heuristic by removing all weights whose relative probability at zero exceeds some threshold $\gamma$, with $0 \leq \gamma \leq 1$. If $q_i(\beta_i)$ is Gaussian this yields

$$\exp\left(-\frac{\mu_i^2}{2\sigma_i^2}\right) > \gamma \implies \left|\frac{\mu_i}{\sigma_i}\right| < \lambda \tag{19}$$

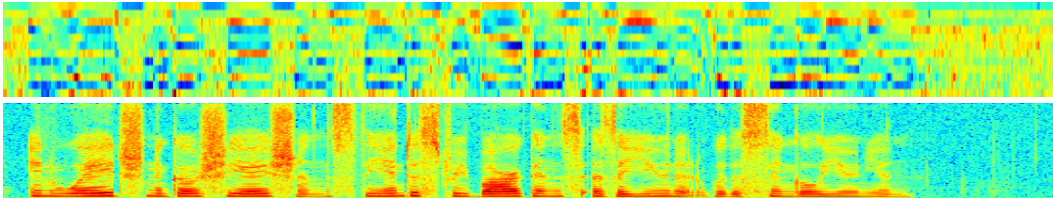

"In wage negotiations the industry bargains as a unit with a single union."

Figure 1: **Two representations of a TIMIT utterance.** Note the lower resolution and greater decorrelation of the MFC coefficients (top) compared to the spectrogram (bottom).

where we have used the reparameterisation $\lambda = \sqrt{-2\ln\gamma}$, with $\lambda \geq 0$. If $\lambda = 0$ no weights are pruned. As $\lambda$ grows the amount of pruning increases, and the probability of the pruned weight vector under $Q(\boldsymbol{\beta})$ (and therefore the likely network performance) decreases. A good rule of thumb for how high $\lambda$ can safely be set is the point at which the pruned weights become less probable than an *average* weight sampled from $q_i(\beta_i)$. For a Gaussian this is

$$\lambda = \sqrt{2\ln\sqrt{2}} \approx 0.83 \tag{20}$$

If the network is retrained after pruning, the cost of transmitting which weights have been removed should in principle be added to $L^C(\boldsymbol{\alpha}, \boldsymbol{\beta})$ (since this information could be used to overfit the training data). However the extra cost does not depend on the network parameters, and can therefore be ignored for the purposes of optimisation.

When a Gaussian prior is used its mean tends to be near zero. This implies that 'cheaper' weights, where $q_i(\beta_i) \approx P(\boldsymbol{\alpha})$, have high relative probability at zero and are thus more likely to be pruned.

## 8   Experiments

We tested all the combinations of posterior and prior described in Section 5 on a hierarchical multidimensional recurrent neural network [9] trained to do phoneme recognition on the TIMIT speech corpus [4]. We also assessed the pruning heuristic from Section 7 by applying it with various thresholds to a trained network and observing the impact on performance and network size.

TIMIT is a popular phoneme recognition benchmark. The core training and test sets (which we used for our experiments) contain respectively 3696 and 192 phonetically transcribed utterances. We defined a validation set by randomly selecting 184 sequences from the training set. The reduced set of 39 phonemes [6] was used during both training and testing. The audio data was presented to the network in the form of spectrogram images. One such image is contrasted with the mel-frequency cepstrum representation used for most speech recognition systems in Fig. 1.

Hierarchical multidimensional recurrent neural networks containing Long Short-Term Memory [11] hidden layers and a CTC output layer [8] have proven effective for offline handwriting recognition [9]. The same architecture is employed here, with a spectrogram in place of a handwriting image, and phoneme labels in place of characters. Since the network scans through the spectrogram in all directions, both vertical and horizontal correlations can be captured.

The network topology was identical for all experiments. It was the same as that of the handwriting recognition network in [9] except that the dimensions of the three subsampling windows used to progressively decrease resolution were now $2 \times 4$, $2 \times 4$ and $1 \times 4$, and the CTC layer now contained 40 output units (one for each phoneme, plus an extra for 'blank'). This gave a total of 15 layers, 1306 units (not counting the inputs or bias), and 139,536 weights. All network parameters were trained with online steepest descent (weight updates after every sequence) using a learning rate of $10^{-4}$ and a momentum of 0.9. For the networks with stochastic derivatives (i.e those with Gaussian posteriors) a single weight sample was drawn for each sequence. Prefix search CTC decoding [8] was used to transcribe the test set, with probability threshold 0.995. When parameters in the posterior or prior were fixed, the best value was found empirically. All networks were initialised with random weights (or random weight means if the posterior was Gaussian), chosen from a Gaussian

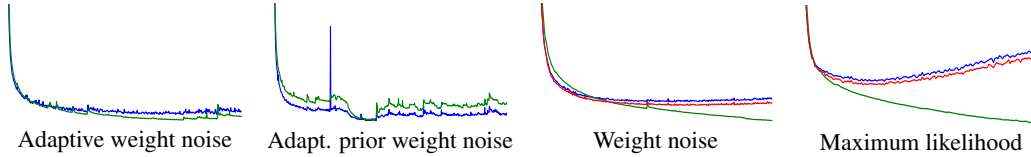

| Adaptive weight noise | Adapt. prior weight noise | Weight noise | Maximum likelihood |

Figure 2: **Error curves for four networks during training.** The green, blue and red curves correspond to the average per-sequence error loss $L^E(\boldsymbol{\beta}, \mathcal{D})$ on the training, test and validation sets respectively. Adaptive weight noise does not overfit, and normal weight noise overfits much more slowly than maximum likelihood. Adaptive weight noise led to longer training times and noisier error curves.

Table 1: **Results for different priors and posteriors.** All distribution parameters were learned by the network unless fixed values are specified. 'Error' is the *phoneme error rate* on the core test set (total edit distance between the network transcriptions and the target transcriptions, multiplied by 100). 'Epochs' is the number of passes through the training set after which the error was recorded. 'Ratio' is the compression ratio of the training set transcription targets relative to a uniform code over the 39 phoneme labels ($\approx 5.3$ bits per phoneme); this could only be calculated for the networks with Gaussian priors and posteriors.

| Name | Posterior | Prior | Error | Epochs | Ratio |
|---|---|---|---|---|---|
| Adaptive L1 | Delta | Laplace | 49.0 | 7 | – |
| Adaptive L2 | Delta | Gauss | 35.1 | 421 | – |
| Adaptive mean L2 | Delta | Gauss $\sigma^2 = 0.1$ | 28.0 | 53 | – |
| L2 | Delta | Gauss $\mu = 0, \sigma^2 = 0.1$ | 27.4 | 59 | – |
| Maximum likelihood | Delta | Uniform | 27.1 | 44 | – |
| L1 | Delta | Laplace $\mu = 0, b = 1/12$ | 26.0 | 545 | – |
| Adaptive mean L1 | Delta | Laplace $b = 1/12$ | 25.4 | 765 | – |
| Weight noise | Gauss $\sigma_i = 0.075$ | Uniform | 25.4 | 220 | – |
| Adaptive prior weight noise | Gauss $\sigma_i = 0.075$ | Gauss | 24.7 | 260 | 0.542 |
| Adaptive weight noise | Gauss | Gauss | 23.8 | 384 | 0.286 |

with mean 0, standard deviation 0.1. For the adaptive Gaussian posterior, the standard deviations of the weights were initialised to 0.075 then optimised during training; this ensured that the variances (which are the standard deviations squared) remained positive. The networks with Gaussian posteriors and priors did not require early stopping and were trained on all 3696 utterances in the training set; all other networks used the validation set for early stopping and hence were trained on 3512 utterances. These were also the only networks for which the transmission cost of the network weights could be measured (since it did not depend on the quantisation of the posterior or prior). The networks were evaluated on the test set using the parameters giving lowest $L^E(\boldsymbol{\beta}, \mathcal{D})$ on the training set (or validation set if present). All experiments were stopped after 100 training epochs with no improvement in either $L(\boldsymbol{\alpha}, \boldsymbol{\beta}, \mathcal{D})$, $L^E(\boldsymbol{\beta}, \mathcal{D})$ or the number of transcription errors on the training or validation set. The reason for such conservative stopping criteria was that the error curves of some of the networks were extremely noisy (see Fig. 2).

Table 1 shows the results for the different posteriors and priors. L2 regularisation was no better than unregularised maximum likelihood, while L1 gave a slight improvement; this is consistent with our previous experience of recurrent neural networks. The fully adaptive L1 and L2 networks performed very badly, apparently because the priors became excessively narrow ($\sigma^2 \approx 0.003$ for L2 and $b \approx 0.002$ for L1). L1 with fixed variance and adaptive mean was somewhat better than L1 with mean fixed at 0 (although the adaptive mean was very close to zero, settling around 0.0064). The networks with Gaussian posteriors outperformed those with delta posteriors, with the best score obtained using a fully adaptive posterior.

Table 2 shows the effect of pruning on the trained 'adaptive weight noise' network from Table 1. The pruned networks were retrained using the same optimisation as before, with the error recorded before and after retraining. As well as being highly effective at removing weights, pruning led to improved performance following retraining in some cases. Notice the slow increase in initial error up to $\lambda = 0.5$ and sharp rise thereafter; this is consistent with the 'safe' threshold of $\lambda \approx 0.83$

Table 2: **Effect of Network Pruning.** '$\lambda$' is the threshold used for pruning. 'Weights' is the number of weights left after pruning and 'Percent' is the same figure expressed as a percentage of the original weights. 'Initial Error' is the test error immediately after pruning and 'Retrain Error' is the test error following 'Retrain Epochs' of subsequent retraining. 'Bits/weight' is the average bit cost (as defined in Eq. (13)) of the *unpruned* weights.

| $\lambda$ | Weights | Percent | Initial error | Retrain error | Retrain Epochs | Bits/weight |
|---|---|---|---|---|---|---|
| 0 | 139,536 | 100% | 23.8 | 23.8 | 0 | 0.53 |
| 0.01 | 107,974 | 77.4% | 23.8 | 24.0 | 972 | 0.72 |
| 0.05 | 63,079 | 45.2% | 23.9 | 23.5 | 35 | 1.15 |
| 0.1 | 52,984 | 37.9% | 23.9 | 23.3 | 351 | 1.40 |
| 0.2 | 43,182 | 30.9% | 23.9 | 23.7 | 740 | 1.82 |
| 0.5 | 31,120 | 22.3% | 24.0 | 23.3 | 125 | 2.21 |
| 1 | 22,806 | 16.3% | 24.5 | 24.1 | 403 | 3.19 |
| 2 | 16,029 | 11.5% | 28.0 | 24.5 | 335 | 3.55 |

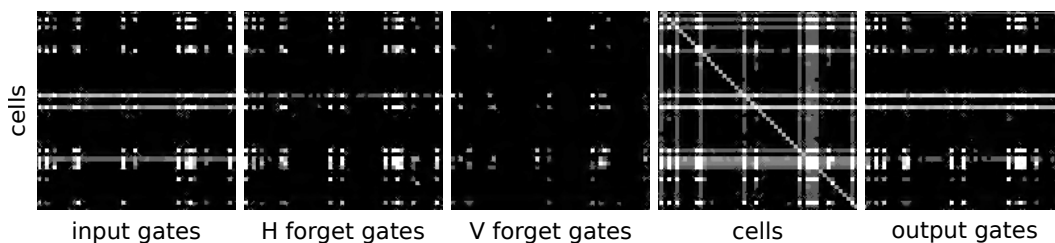

input gates     H forget gates     V forget gates     cells     output gates

Figure 3: **Weight costs in an 2D LSTM recurrent connection.** Each dot corresponds to a weight; the lighter the colour the more bits the weight costs. The vertical axis shows the LSTM cell the weight comes *from*; the horizontal axis shows the LSTM unit the weight goes *to*. Note the low cost of the 'V forget gates' (these mediate vertical correlations between frequency bands in the spectrogram, which are apparently less important to transcription than horizontal correlations between timesteps); the high cost of the 'cells' (LSTM's main processing units); the bright horizontal and vertical bands (corresponding to units with 'important' outputs and inputs respectively); and the bright diagonal through the cells (corresponding to self connections).

mentioned in Section 7. The lowest final phoneme error rate of 23.3 would until recently have been the best recorded on TIMIT; however the application of deep belief networks has now improved the benchmark to 20.5 [3].

**Acknowledgements**

I would like to thank Geoffrey Hinton, Christian Osendorfer, Justin Bayer and Thomas Rückstieß for helpful discussions and suggestions. Alex Graves is a Junior Fellow of the Canadian Institute for Advanced Research.

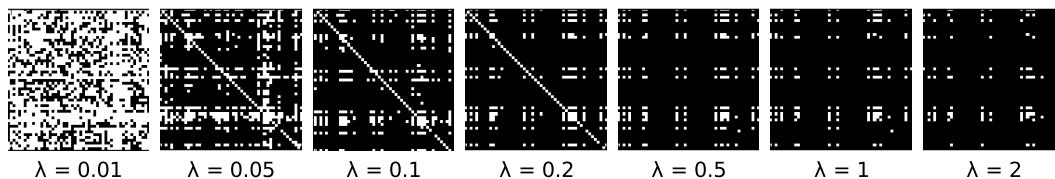

$\lambda = 0.01$    $\lambda = 0.05$    $\lambda = 0.1$    $\lambda = 0.2$    $\lambda = 0.5$    $\lambda = 1$    $\lambda = 2$

Figure 4: **The 'cell' weights from Fig. 3 pruned at different thresholds.** Black dots are pruned weights, white dots are remaining weights. 'Cheaper' weights tend to be removed first as $\lambda$ grows.

## Footnotes

[1]An important exception are energy-based models such as restricted Boltzmann machines [24] whose log-loss is intractable.

[2]Unsupervised learning can be treated as a special case where $\mathbf{x} = \emptyset$

[3]The floating point resolution of the computer architecture used to train the network could in principle be used to upper-bound the discretisation constant, and hence the compression; but in practice the bound would be prohibitively high.

# References

[1] D. Barber and C. M. Bishop. *Ensemble learning in Bayesian neural networks.*, pages 215–237. Springer-Verlag, Berlin, 1998.

[2] D. Barber and B. Schottky. Radial basis functions: A bayesian treatment. In *NIPS*, 1997.

[3] G. E. Dahl, M. Ranzato, A. rahman Mohamed, and G. Hinton. Phone recognition with the mean-covariance restricted boltzmann machine. In J. Lafferty, C. K. I. Williams, J. Shawe-Taylor, R. Zemel, and A. Culotta, editors, *Advances in Neural Information Processing Systems 23*, pages 469–477. 2010.

[4] DARPA-ISTO. *The DARPA TIMIT Acoustic-Phonetic Continuous Speech Corpus (TIMIT)*, speech disc cd1-1.1 edition, 1990.

[5] B. J. Frey. *Graphical models for machine learning and digital communication*. MIT Press, Cambridge, MA, USA, 1998.

[6] K. fu Lee and H. wuen Hon. Speaker-independent phone recognition using hidden markov models. *IEEE Transactions on Acoustics, Speech, and Signal Processing*, 1989.

[7] C. L. Giles and C. W. Omlin. Pruning recurrent neural networks for improved generalization performance. *IEEE Transactions on Neural Networks*, 5:848–851, 1994.

[8] A. Graves, S. Fernández, F. Gomez, and J. Schmidhuber. Connectionist temporal classification: Labelling unsegmented sequence data with recurrent neural networks. In *Proceedings of the International Conference on Machine Learning, ICML 2006*, Pittsburgh, USA, 2006.

[9] A. Graves and J. Schmidhuber. Offline handwriting recognition with multidimensional recurrent neural networks. In *NIPS*, pages 545–552, 2008.

[10] G. E. Hinton and D. van Camp. Keeping the neural networks simple by minimizing the description length of the weights. In *COLT*, pages 5–13, 1993.

[11] S. Hochreiter and J. Schmidhuber. Long Short-Term Memory. *Neural Computation*, 9(8):1735–1780, 1997.

[12] A. Honkela and H. Valpola. Variational learning and bits-back coding: An information-theoretic view to bayesian learning. *IEEE Transactions on Neural Networks*, 15:800–810, 2004.

[13] K.-C. Jim, C. Giles, and B. Horne. An analysis of noise in recurrent neural networks: convergence and generalization. *Neural Networks, IEEE Transactions on*, 7(6):1424 –1438, nov 1996.

[14] N. D. Lawrence. *Variational Inference in Probabilistic Models*. PhD thesis, University of Cambridge, 2000.

[15] Y. Le Cun, J. Denker, and S. Solla. Optimal brain damage. In D. S. Touretzky, editor, *Advances in Neural Information Processing Systems*, volume 2, pages 598–605. Morgan Kaufmann, San Mateo, CA, 1990.

[16] D. J. C. MacKay. Probable networks and plausible predictions - a review of practical bayesian methods for supervised neural networks. *Neural Computation*, 1995.

[17] S. J. Nowlan and G. E. Hinton. Simplifying neural networks by soft weight sharing. *Neural Computation*, 4:173–193, 1992.

[18] M. Opper and C. Archambeau. The variational gaussian approximation revisited. *Neural Computation*, 21(3):786–792, 2009.

[19] D. Plaut, S. Nowlan, and G. E. Hinton. Experiments on learning by back propagation. Technical Report CMU-CS-86-126, Department of Computer Science, Carnegie Mellon University, Pittsburgh, PA, 1986.

[20] M. Riedmiller and T. Braun. A direst adaptive method for faster backpropagation learning: The rprop algorithm. In *International Symposium on Neural Networks*, 1993.

[21] J. Rissanen. Modeling by shortest data description. *Automatica*, 14(5):465 – 471, 1978.

[22] D. E. Rumelhart, G. E. Hinton, and R. J. Williams. *Learning representations by back-propagating errors*, pages 696–699. MIT Press, Cambridge, MA, USA, 1988.

[23] C. E. Shannon. A mathematical theory of communication. *Bell system technical journal*, 27, 1948.

[24] P. Smolensky. *Information processing in dynamical systems: foundations of harmony theory*, pages 194–281. MIT Press, Cambridge, MA, USA, 1986.

[25] C. S. Wallace. Classification by minimum-message-length inference. In *Proceedings of the international conference on Advances in computing and information*, ICCI'90, pages 72–81, New York, NY, USA, 1990. Springer-Verlag New York, Inc.

[26] I. H. Witten, R. M. Neal, and J. G. Cleary. Arithmetic coding for data compression. *Commun. ACM*, 30:520–540, June 1987.

